# Contextual Modulation of Target Saliency

**Antonio Torralba**
Dept. of Brain and Cognitive Sciences
MIT, Cambridge, MA 02139
*torralba@ai.mit.edu*

## Abstract

The most popular algorithms for object detection require the use of exhaustive spatial and scale search procedures. In such approaches, an object is defined by means of local features. In this paper we show that including contextual information in object detection procedures provides an efficient way of cutting down the need for exhaustive search. We present results with real images showing that the proposed scheme is able to accurately predict likely object classes, locations and sizes.

## 1  Introduction

Although there is growing evidence of the role of contextual information in human perception [1], research in computational vision is dominated by object-based representations [5,9,10,15]. In real-world scenes, intrinsic object information is often degraded due to occlusion, low contrast, and poor resolution. In such situations, the object recognition problem based on intrinsic object representations is ill-posed. A more comprehensive representation of an object should include contextual information [11,13]: *Obj. representation = {intrisic obj. model, contextual obj. model}*. In this representation, an object is defined by 1) a model of the intrinsic properties of the object and 2) a model of the typical contexts in which the object is immersed. Here we show how incorporating contextual models can enhance target object saliency and provide an estimate of its likelihood and intrinsic properties.

## 2  Target saliency and object likelihood

Image information can be partitioned into two sets of features: *local features*, $\vec{v}_L$, that are intrinsic to an object, and *contextual features*, $\vec{v}_c$ which encode structural properties of the background. In a statistical framework, object detection requires evaluation of the likelihood function (target saliency function): $P(O \,|\, \vec{v}_L, \vec{v}_C)$ which provides the probability of presence of the object $O$ given a set of local and contextual measurements. $O$ is the set of parameters that define an object immersed in a scene: $O = \{o_n, x, y, \vec{t}\}$ with $o_n$=object class, (x,y)=location in image coordinates

and $\vec{t}$=object appearance parameters. By applying Bayes rule we can write:

$$P(O \,|\, \vec{v}_L, \vec{v}_C) = \frac{1}{P(\vec{v}_L \,|\, \vec{v}_C)} P(\vec{v}_L \,|\, O, \vec{v}_C) P(O \,|\, \vec{v}_C) \tag{1}$$

Those three factors provide a simplified framework for representing three levels of attention guidance when looking for a target: The normalization factor, $1/P(\vec{v}_L \,|\, \vec{v}_C)$, does not depend on the target or task constraints, and therefore is a bottom-up factor. It provides a measure of how unlikely it is to find a set of local measurements $\vec{v}_L$ within the context $\vec{v}_C$. We can define local saliency as $S(x,y) = 1/P(\vec{v}_L(x,y) \,|\, \vec{v}_C)$. Saliency is large for unlikely features in a scene. The second factor, $P(\vec{v}_L \,|\, O, \vec{v}_C)$, gives the likelihood of the local measurements $\vec{v}_L$ when the object is present at such location in a particular context. We can write $P(\vec{v}_L \,|\, O, \vec{v}_C) \simeq P(\vec{v}_L \,|\, O)$, which is a convenient approximation when the aspect of the target object is fully determined by the parameters given by the description O. This factor represents the top-down knowledge of the target appearance and how it contributes to the search. Regions of the image with features unlikely to belong to the target object are vetoed and regions with attended features are enhanced. The third factor, the PDF $P(O \,|\, \vec{v}_C)$, provides context-based priors on object class, location and scale. It is of capital importance for insuring reliable inferences in situations where the local image measurements $\vec{v}_L$ produce ambiguous interpretations. This factor does not depend on local measurements and target models [8,13]. Therefore, the term $P(O \,|\, \vec{v}_C)$ modulates the saliency of local image properties when looking for an object of the class $o_n$. Contextual priors become more evident if we apply Bayes rule successively in order to split the PDF $P(O \,|\, \vec{v}_C)$ into three factors that model three kinds of context priming on object search:

$$P(O, \,|\, \vec{v}_C) \simeq P(\vec{t} \,|\, \vec{v}_C, o_n) P(x, y \,|\, \vec{v}_C, o_n) P(o_n, \,|\, \vec{v}_C) \tag{2}$$

According to this decomposition of the PDF, the contextual modulation of target saliency is a function of three main factors:

*Object likelihood*: $P(o_n \,|\, \vec{v}_C)$ provides the probability of presence of the object class $o_n$ in the scene. If $P(o_n \,|\, \vec{v}_C)$ is very small, then object search need not be initiated (we do not need to look for cars in a living room).

*Contextual control of focus of attention*: $P(x, y \,|\, o_n, \vec{v}_C)$. This PDDF gives the most likely locations for the presence of object $o_n$ given context information, and it allocates computational resources into relevant scene regions.

*Contextual selection of local target appearance*: $P(\vec{t} \,|\, \vec{v}_C, o_n)$. This gives the likely (prototypical) shapes (point of views, size, aspect ratio, object aspect) of the object $o_n$ in the context $\vec{v}_C$. Here $\vec{t} = \{\sigma, p\}$, with $\sigma$=scale and $p$=aspect ratio. Other parameters describing the appearance of an object in an image can be added.

The image features most commonly used for describing local structures are the energy outputs of oriented band-pass filters, as they have been shown to be relevant for the task of object detection [9,10] and scene recognition [2,4,8,12]. Therefore, the local image representation at the spatial location ($\vec{x}$) is given by the vector $\vec{v}_L(\vec{x}) = \{v(\vec{x}, k)\}_{k=1,N}$ with:

$$v(\vec{x}, k) = \left| \sum_{\vec{x}'} i(\vec{x}') g_k(\vec{x} - \vec{x}') \right| \tag{3}$$

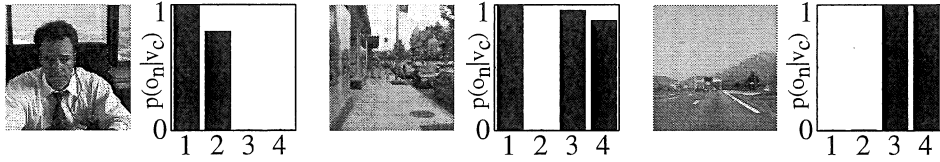

Figure 1: Contextual object priming of four objects categories (1-people, 2-furniture, 3-vehicles and 4-trees)

where $i(\vec{x})$ is the input image and $g_k(\vec{x})$ are oriented band-pass filters defined by $g_k(\vec{x}) = e^{-\|\vec{x}\|^2/\sigma_k^2}e^{2\pi j<\vec{f_k},\vec{x}>}$. In such a representation [8], $v(\vec{x}, k)$ is the output magnitude at the location $\vec{x}$ of a complex Gabor filter tuned to the spatial frequency $\vec{f_k}$. The variable $k$ indexes filters tuned to different spatial frequencies and orientations.

On the other hand, contextual features have to summarize the structure of the whole image. It has been shown that a holistic low-dimensional encoding of the local image features conveys enough information for a semantic categorization of the scene/context [8] and can be used for contextual priming in object recognition tasks [13]. Such a representation can be achieved by decomposing the image features into the basis functions provided by PCA:

$$a_n = \sum_{\vec{x}}\sum_{k} v(\vec{x}, k)\,\psi_n(\vec{x}, k) \qquad v(\vec{x}, k) \simeq \sum_{n=1}^{N} a_n\psi_n(\vec{x}, k) \qquad (4)$$

We propose to use the decomposition coefficients $\vec{v}_C = \{a_n\}_{n=1,N}$ as context features. The functions $\psi_n$ are the eigenfunctions of the covariance operator given by $v(\vec{x}, k)$. By using only a reduced set of components ($N = 60$ for the rest of the paper), the coefficients $\{a_n\}_{n=1,N}$ encode the main spectral characteristics of the scene with a coarse description of their spatial arrangement. In essence, $\{a_n\}_{n=1,N}$ is a holistic representation as all the regions of the image contribute to all the coefficients, and objects are not encoded individually [8]. In the rest of the paper we show the efficacy of this set of features in context modeling for object detection tasks.

## 3  Contextual object priming

The PDF $P(o_n\,|\,\vec{v}_C)$ gives the probability of presence of the object class $o_n$ given contextual information. In other words, the PDF $P(o_n\,|\,\vec{v}_C)$ evaluates the consistency of the object $o_n$ with the context $\vec{v}_C$. For instance, a car has a high probability of presence in a highway scene but it is inconsistent with an indoor environment. The goal of $P(o_n\,|\,\vec{v}_C)$ is to cut down the number of possible object categories to deal with before expending computational resources in the object recognition process. The learning of the PDF $P(o_n\,|\,\vec{v}_C) = P(\vec{v}_C\,|\,o_n)P(o_n)/p(\vec{v}_C)$ with $p(\vec{v}_C) = P(\vec{v}_C\,|\,o_n)P(o_n) + P(\vec{v}_C\,|\,\neg o_n)P(\neg o_n)$ is done by approximating the in-class and out-of-class PDFs by a mixture of Gaussians:

$$P(\vec{v}_C\,|\,o_n) = \sum_{i=1}^{L} b_{i,n}G(\vec{v}_C; \vec{v}_{i,n}, \mathbf{V}_{i,n}) \qquad (5)$$

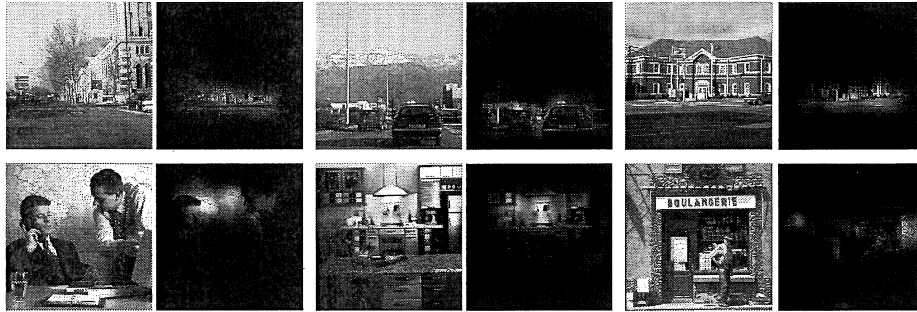

Figure 2: Contextual control of focus of attention when the algorithm is looking for cars (upper row) or heads (bottom row).

The model parameters $(b_{i,n}, \vec{v}_{i,n}, \mathbf{V}_{i,n})$ for the object class $o_n$ are obtained using the EM algorithm [3]. The learning requires the use of few Gaussian clusters ($L = 2$ provides very good performances). For the learning, the system is trained with a set of examples manually annotated with the presence/absence of four objects categories (1-people, 2-furniture, 3-vehicles and 4-trees). Fig. 1 shows some typical results from the priming model on the four superordinate categories of objects defined. Note that the probability function $P(o_n \mid \vec{v}_C)$ provides information about the probable presence of one object without scanning the picture. If $P(o_n \mid \vec{v}_C) > 1 - th$ then we can predict that the target is present. On the other hand, if $P(o_n \mid \vec{v}_C) < th$ we can predict that the object is likely to be absent before exploring the image.

The number of scenes in which the system may be able to take high confidence decisions will depend on different factors such as: the strength of the relationship between the target object and its context and the ability of $\vec{v}_C$ for efficiently characterizing the context. Figure 1 shows some typical results from the priming model for a set of super-ordinate categories of objects. When forcing the model to take binary decisions in all the images (by selecting an acceptance threshold of $th = 0.5$) the presence/absence of the objects was correctly predicted by the model on 81% of the scenes of the test set. For each object category, high confidence predictions ($th = .1$) were made in at least 50% of the tested scene pictures and the presence/absence of each object class was correctly predicted by the model on 95% of those images. Therefore, for those images, we do not need to use local image analysis to decide about the presence/absence of the object.

## 4  Contextual control of focus of attention

One of the strategies that biological visual systems use to deal with the analysis of real-world scenes is to focus attention (and, therefore, computational resources) onto the important image regions while neglecting others. Current computational models of visual attention (saliency maps and target detection) rely exclusively on local information or intrinsic object models [6,7,9,14,16]. The control of the focus of attention by contextual information that we propose here is both task driven (looking for object $o_n$) and context driven (given global context information: $\vec{v}_C$). However, it does not include any model of the target object at this stage. In our framework, the problem of contextual control of the focus of attention involves the

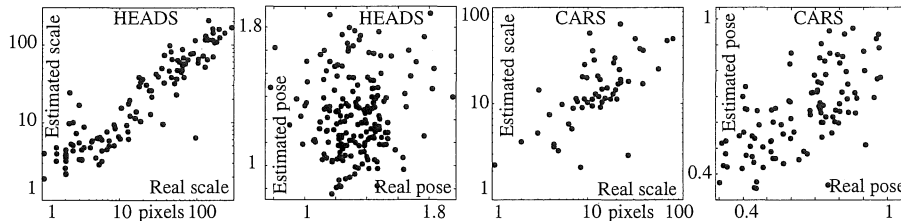

Figure 3: Estimation results of object *scale* and *pose* based on contextual features.

evaluation of the PDF $P(\vec{x}|o_n,\vec{v}_C)$. For the learning, the joint PDF is modeled as a sum of gaussian clusters. Each cluster is decomposed into the product of two gaussians modeling respectively the distribution of object locations and the distribution of contextual features for each cluster:

$$P(\vec{x},\vec{v}_C|o_n) = \sum_{i=1}^{L} b_{i,n}\, G(\vec{x};\vec{x}_{i,n},\mathbf{X}_{i,n})G(\vec{v}_C;\vec{v}_{i,n},\mathbf{V}_{i,n}) \qquad (6)$$

The training set used for the learning of the PDF $P(\vec{x},\vec{v}_C|o_n)$ is a subset of the pictures that contain the object $o_n$. The training data is $\{\vec{v}_t\}_{t=1,N_t}$ and $\{\vec{x}_t\}_{t=1,N_t}$ where $\vec{v}_t$ are the contextual features of the picture $t$ of the training set and $\vec{x}_t$ is the location of object $o_n$ in the image. The model parameters are obtained using the EM algorithm [3,13]. We used 1200 pictures for training and a separate set of 1200 pictures for testing. The success of the PDF in narrowing the region of the focus of attention will depend on the consistency of the relationship between the object and the context. Fig. 2 shows several examples of images and the selected regions based on contextual features when looking for cars and faces. From the PDF $P(\vec{x},\vec{v}_C|o_n)$ we selected the region with the highest probability (33% of the image size on average). 87% of the heads present in the test pictures were inside the selected regions.

## 5   Contextual selection of object appearance models

One major problem for computational approaches to object detection is the large variability in object appearance. The classical solution is to explore the space of possible shapes looking for the best match. The main sources of variability in object appearance are size, pose and intra-class shape variability (deformations, style, etc.). We show here that including contextual information can reduce at least the first two sources of variability. For instance, the expected size of people in an image differs greatly between an indoor environment and a perspective view of a street. Both environments produce different patterns of contextual features $\vec{v}_C$ [8]. For the second factor, pose, in the case of cars, there is a strong relationship between the possible orientations of the object and the scene configuration. For instance, looking down a highway, we expect to see the back of the cars, however, in a street view, looking towards the buildings, lateral views of cars are more likely.

The expected scale and pose of the target object can be estimated by a regression procedure. The training database used for building the regression is a set of 1000 images in which the target object $o_n$ is present. For each training image the target

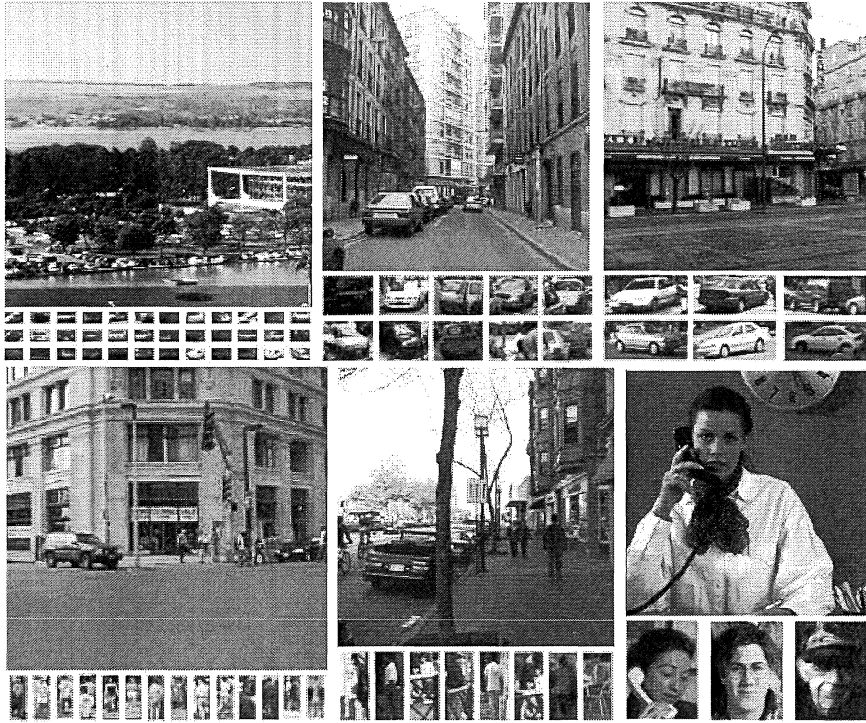

Figure 4: Selection of prototypical object appearances based on contextual cues.

object was selected by cropping a rectangular window. For faces and cars we define the $\sigma = scale$ as the height of the selected window and the $p = pose$ as the ratio between the horizontal and vertical dimensions of the window $(\Delta y / \Delta x)$. On average, this definition of pose provides a good estimation of the orientation for cars but not for heads. Here we used regression using a mixture of gaussians for estimating the conditional PDFs between scale, pose and contextual features: $P(\sigma \,|\, \vec{v}_C, \, o_n)$ and $P(p \,|\, \vec{v}_C, \, o_n)$. This yields the next regression procedures [3]:

$$\bar{\sigma} = \frac{\sum_i \sigma_{i,n} b_{i,n} G(\vec{v}_C; \vec{v}_{i,n}, \mathbf{V}_{i,n})}{\sum_i b_{i,n} G(\vec{v}_C; \vec{v}_{i,n}, \mathbf{V}_{i,n})} \qquad \bar{p} = \frac{\sum_i p_{i,n} b_{i,n} G(\vec{v}_C; \vec{v}_{i,n}, \mathbf{V}_{i,n})}{\sum_i b_{i,n} G(\vec{v}_C; \vec{v}_{i,n}, \mathbf{V}_{i,n})} \qquad (7)$$

The results summarized in fig. 3 show that context is a strong cue for scale selection for the face detection task but less important for the car detection task. On the other hand, context introduces strong constraints on the prototypical point of views of cars but not at all for heads. Once the two parameters (pose and scale) have been estimated, we can build a prototypical model of the target object. In the case of a view-based object representation, the model of the object will consist of a collection of templates that correspond to the possible aspects of the target. For each image the system produces a collection of views, selected among a database of target examples that have the scale and pose given by eqs. (7). Fig. 4 shows some results from this procedure. In the statistical framework, the object detection requires the evaluation of the function $P(\vec{v}_L \,|\, O, \, \vec{v}_C)$. We can approximate

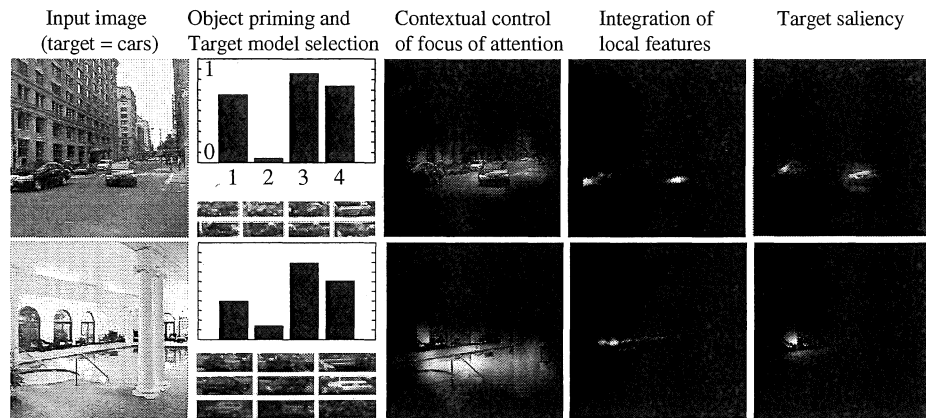

Figure 5: Schematic layout of the model for object detection (here cars) by integration of contextual and local information. The bottom example is an error in detection due to incorrect context identification.

$P(\vec{v}_L \mid O, \vec{v}_C) \simeq P(\vec{v}_L \mid o_n, \sigma, p)$. Fig. 5 and 6 show the complete chain of operations and some detection results using a simple correlation technique between the image and the generated object models (100 exemplars) at only one scale. The last image of each row shows the total object likelihood obtained by multiplying the object saliency maps (obtained by the correlation) and the contextual control of the focus of attention. The result shows how the use of context helps reduce false alarms. This results in good detection performances despite the simplicity of the matching procedure used.

## 6    Conclusion

The contextual schema of a scene provides the likelihood of presence, typical locations and appearances of objects within the scene. We have proposed a model for incorporating such contextual cues in the task of object detection. The main aspects of our approach are: 1) Progressive reduction of the window of focus of attention: the system reduces the size of the focus of attention by first integrating contextual information and then local information. 2) Inhibition of target like patterns that are in inconsistent locations. 3) Faster detection of correctly scaled targets that have a pose in agreement with the context. 4) No requirement of parsing a scene into individual objects. Furthermore, once one object has been detected, it can introduce new contextual information for analyzing the rest of the scene.

**Acknowledgments**

The author wishes to thank Dr. Pawan Sinha, Dr. Aude Oliva and Prof. Whitman Richards for fruitful discussions.

**References**

[1] Biederman, I., Mezzanotte, R.J., & Rabinowitz, J.C. (1982). Scene perception: detecting and judging objects undergoing relational violations. *Cognitive Psychology*, 14:143–177.

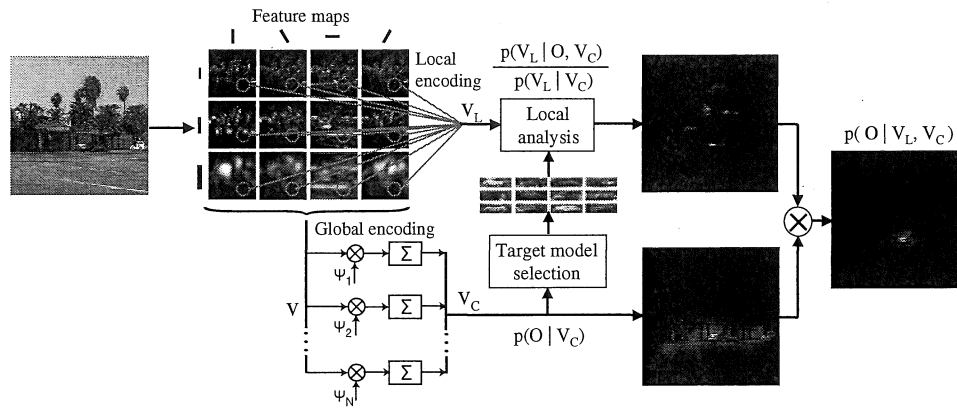

Figure 6: Schema for object detection (e.g. cars) integrating local and global information.

[2] Carson, C., Belongie, S., Greenspan, H., and Malik, J. (1997). Region-based image querying. *Proc. IEEE W. on Content-Based Access of Image and Video Libraries*, pp: 42–49.

[3] Gershnfeld, N. *The nature of mathematical modeling.* Cambridge university press, 1999.

[4] Gorkani, M. M., Picard, R. W. (1994). Texture orientation for sorting photos 'at a glance'. Proc. Int. Conf. Pat. Rec., Jerusalem, Vol. I: 459-464.

[5] Heisle, B., T. Serre, S. Mukherjee and T. Poggio. (2001) Feature Reduction and Hierarchy of Classifiers for Fast Object Detection in Video Images. In: Proceedings of 2001 IEEE Computer Society Conference on Computer Vision and Pattern Recognition, IEEE Computer Society Press, Jauai, Hawaii.

[6] Itti, L., Koch, C., & Niebur, E. (1998). A model of saliency-based visual attention for rapid scene analysis. *IEEE Trans. Pattern Analysis and Machine Vision*, 20(11):1254.

[7] Moghaddam, B., & Pentland, A. (1997). Probabilistic Visual Learning for Object Representation. *IEEE Trans. Pattern Analysis and Machine Vision*, 19(7):696-710.

[8] Oliva, A., & Torralba, A. (2001). Modeling the Shape of the Scene: A holistic representation of the spatial envelope. *Int. Journal of Computer Vision*, 42(3):145-175.

[9] Rao, R.P.N., Zelinsky, G.J., Hayhoe, M.M., & Ballard, D.H. (1996). Modeling saccadic targeting in visual search. *NIPS 8.* Cambridge, MA: MIT Press.

[10] Schiele, B., Crowley, J. L. (2000) Recognition without Correspondence using Multidimensional Receptive Field Histograms, Int. Journal of Computer Vision, Vol. 36(1):31-50.

[11] Strat, T. M., & Fischler, M. A. (1991). Context-based vision: recognizing objects using information from both 2-D and 3-D imagery. *IEEE trans. on Pattern Analysis and Machine Intelligence*, 13(10): 1050-1065.

[12] Szummer, M., and Picard, R. W. (1998). Indoor-outdoor image classification. In *IEEE intl. workshop on Content-based Access of Image and Video Databases*, 1998.

[13] Torralba, A., & Sinha, P. (2001). Statistical context priming for object detection. *IEEE Proc. Of Int. Conf in Comp. Vision.*

[14] Treisman, A., & Gelade, G. (1980). A feature integration theory of attention. *Cognitive Psychology*, Vol. 12:97–136.

[15] Viola, P. and Jones, M. (2001). Rapid object detection using a boosted cascade of simple features. In: Proceedings of 2001 IEEE Computer Society Conference on Computer Vision and Pattern Recognition (CVPR 2001), IEEE Computer Society Press, Jauai, Hawaii.

[16] Wolfe, J. M. (1994). Guided search 2.0. A revised model of visual search. *Psychonomic Bulletin and Review*, 1:202-228
